# Handwritten Digit Recognition with a Back-Propagation Network

Y. Le Cun, B. Boser, J. S. Denker, D. Henderson,
R. E. Howard, W. Hubbard, and L. D. Jackel
AT&T Bell Laboratories, Holmdel, N. J. 07733

## ABSTRACT

We present an application of back-propagation networks to handwritten digit recognition. Minimal preprocessing of the data was required, but architecture of the network was highly constrained and specifically designed for the task. The input of the network consists of normalized images of isolated digits. The method has 1% error rate and about a 9% reject rate on zipcode digits provided by the U.S. Postal Service.

## 1   INTRODUCTION

The main point of this paper is to show that large back-propagation (BP) networks can be applied to real image-recognition problems without a large, complex preprocessing stage requiring detailed engineering. Unlike most previous work on the subject (Denker et al., 1989), the learning network is directly fed with images, rather than feature vectors, thus demonstrating the ability of BP networks to deal with large amounts of low level information.

Previous work performed on simple digit images (Le Cun, 1989) showed that the architecture of the network strongly influences the network's generalization ability. Good generalization can only be obtained by designing a network architecture that contains a certain amount of *a priori* knowledge about the problem. The basic design principle is to minimize the number of free parameters that must be determined by the learning algorithm, without overly reducing the computational power of the network. This principle increases the probability of correct generalization because

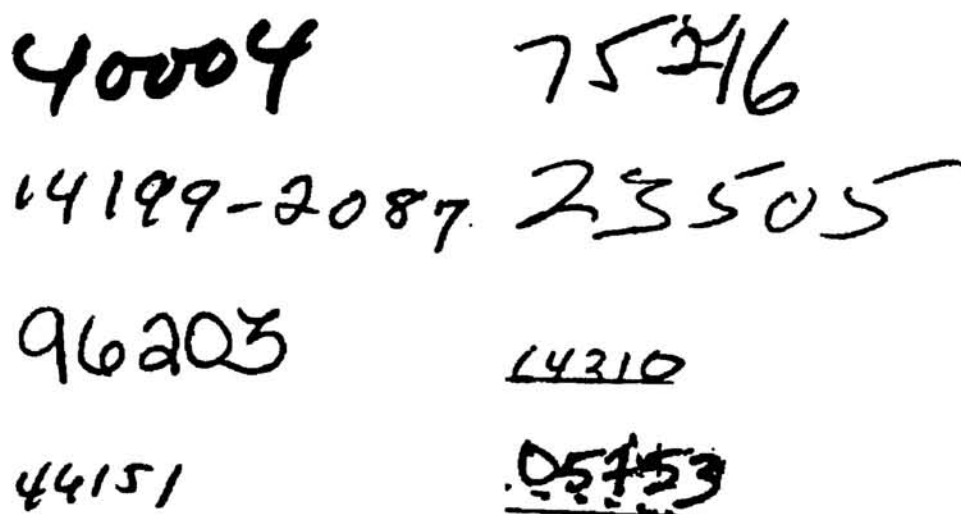

**Figure 1:** Examples of original zipcodes from the testing set.

it results in a specialized network architecture that has a reduced entropy (Denker et al., 1987; Patarnello and Carnevali, 1987; Tishby, Levin and Solla, 1989; Le Cun, 1989). On the other hand, some effort must be devoted to designing appropriate constraints into the architecture.

## 2 ZIPCODE RECOGNITION

The handwritten digit-recognition application was chosen because it is a relatively simple machine vision task: the input consists of black or white pixels, the digits are usually well-separated from the background, and there are only ten output categories. Yet the problem deals with objects in a real two-dimensional space and the mapping from image space to category space has both considerable regularity and considerable complexity. The problem has added attraction because it is of great practical value.

The database used to train and test the network is a superset of the one used in the work reported last year (Denker et al., 1989). We emphasize that the method of solution reported here relies more heavily on automatic learning, and much less on hand-designed preprocessing.

The database consists of 9298 segmented numerals digitized from handwritten zipcodes that appeared on real U.S. Mail passing through the Buffalo, N.Y. post office. Examples of such images are shown in figure 1. The digits were written by many different people, using a great variety of sizes, writing styles and instruments, with widely varying levels of care. This was supplemented by a set of 3349 printed digits coming from 35 different fonts. The training set consisted of 7291 handwritten digits plus 2549 printed digits. The remaining 2007 handwritten and 700 printed digits were used as the test set. The printed fonts in the test set were different from the printed fonts in the training set.One important feature of this database, which

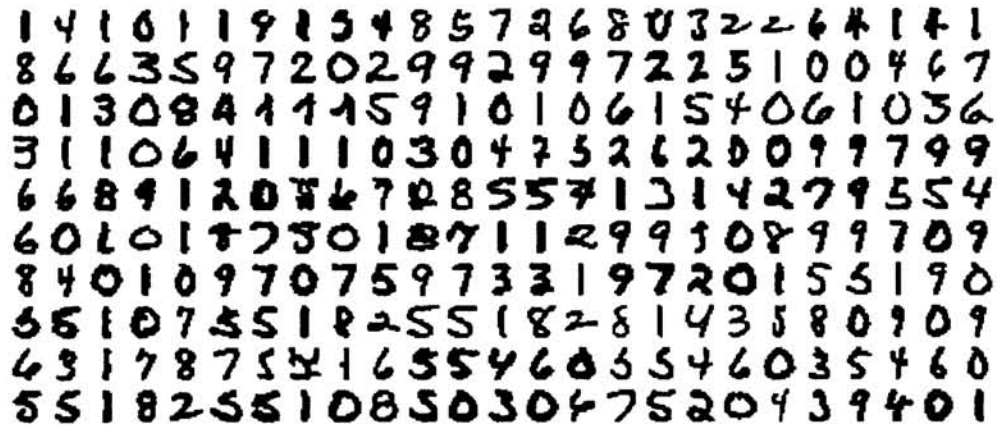

**Figure 2:** Examples of normalized digits from the testing set.

is a common feature to all real-world databases, is that both the training set and the testing set contain numerous examples that are ambiguous, unclassifiable, or even misclassified.

## 3   PREPROCESSING

Acquisition, binarization, location of the zipcode, and preliminary segmentation were performed by Postal Service contractors (Wang and Srihari, 1988). Some of these steps constitute very hard tasks in themselves. The segmentation (separating each digit from its neighbors) would be a relatively simple task if we could assume that a character is contiguous and is disconnected from its neighbors, but neither of these assumptions holds in practice. Many ambiguous characters in the database are the result of mis-segmentation (especially broken 5's) as can be seen on figure 2.

At this point, the size of a digit varies but is typically around 40 by 60 pixels. Since the input of a back-propagation network is fixed size, it is necessary to normalize the size of the characters. This was performed using a linear transformation to make the characters fit in a 16 by 16 pixel image. This transformation preserves the aspect ratio of the character, and is performed after extraneous marks in the image have been removed. Because of the linear transformation, the resulting image is not binary but has multiple gray levels, since a variable number of pixels in the original image can fall into a given pixel in the target image. The gray levels of each image are scaled and translated to fall within the range −1 to 1.

## 4   THE NETWORK

The remainder of the recognition is entirely performed by a multi-layer network. All of the connections in the network are adaptive, although heavily constrained, and are trained using back-propagation. This is in contrast with earlier work (Denker et al., 1989) where the first few layers of connections were hand-chosen constants. The input of the network is a 16 by 16 normalized image and the output is composed

of 10 units: one per class. When a pattern belonging to class $i$ is presented, the desired output is +1 for the $i$th output unit, and −1 for the other output units.

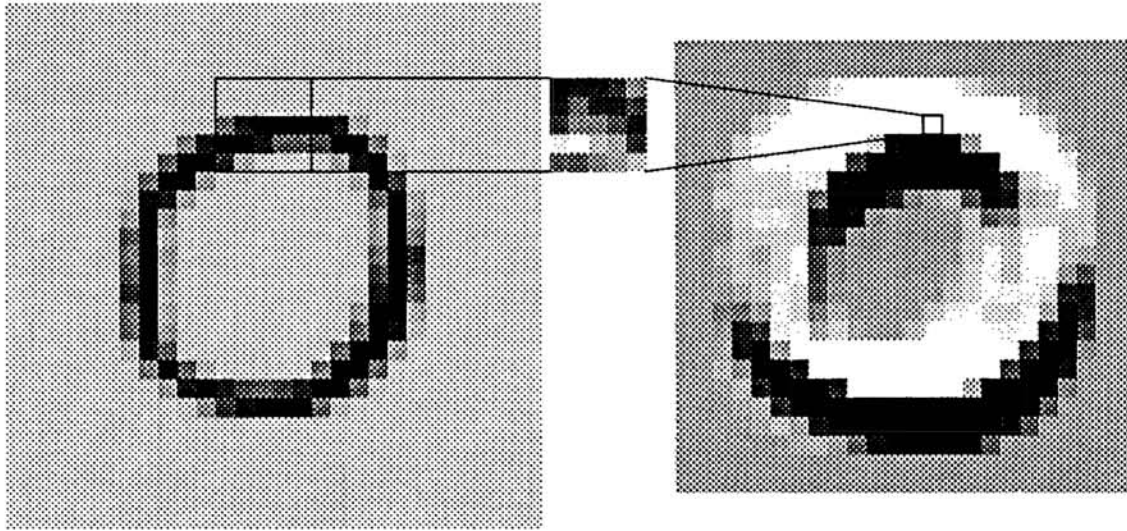

**Figure 3:** Input image (left), weight vector (center), and resulting feature map (right). The feature map is obtained by scanning the input image with a single neuron that has a local receptive field, as indicated. White represents -1, black represents +1.

A fully connected network with enough discriminative power for the task would have far too many parameters to be able to generalize correctly. Therefore a restricted connection-scheme must be devised, guided by our prior knowledge about shape recognition. There are well-known advantages to performing shape recognition by detecting and combining local features. We have required our network to do this by constraining the connections in the first few layers to be local. In addition, if a feature detector is useful on one part of the image, it is likely to be useful on other parts of the image as well. One reason for this is that the salient features of a distorted character might be displaced slightly from their position in a typical character. One solution to this problem is to scan the input image with a single neuron that has a local receptive field, and store the states of this neuron in corresponding locations in a layer called a *feature map* (see figure 3). This operation is equivalent to a convolution with a small size kernel, followed by a squashing function. The process can be performed in parallel by implementing the feature map as a plane of neurons whose weight vectors are constrained to be equal. That is, units in a feature map are constrained to perform the same operation on different parts of the image. An interesting side-effect of this *weight sharing* technique, already described in (Rumelhart, Hinton and Williams, 1986), is to reduce the number of free parameters by a large amount, since a large number of units share the same weights. In addition, a certain level of shift invariance is present in the system: shifting the input will shift the result on the feature map, but will leave it unchanged otherwise. In practice, it will be necessary to have multiple feature maps, extracting different features from the same image.

|   | 1 | 2 | 3 | 4 | 5 | 6 | 7 | 8 | 9 | 10 | 11 | 12 |
|---|---|---|---|---|---|---|---|---|---|----|----|----|
| 1 | X | X | X |   | X | X |   |   |   |    |    |    |
| 2 |   | X | X | x | X | X |   |   |   |    |    |    |
| 3 |   |   |   |   |   |   | X | X | X |    | X  | X  |
| 4 |   |   |   |   |   |   |   | X | X | x  | X  | X  |

**Table 1:** Connections between H2 and H3.

The idea of local, convolutional feature maps can be applied to subsequent hidden layers as well, to extract features of increasing complexity and abstraction. Interestingly, higher level features require less precise coding of their location. Reduced precision is actually advantageous, since a slight distortion or translation of the input will have reduced effect on the representation. Thus, each feature extraction in our network is followed by an additional layer which performs a local averaging and a subsampling, reducing the resolution of the feature map. This layer introduces a certain level of invariance to distortions and translations. A functional module of our network consists of a layer of shared-weight feature maps followed by an averaging/subsampling layer. This is reminiscent of the Neocognitron architecture (Fukushima and Miyake, 1982), with the notable difference that we use backprop (rather than unsupervised learning) which we feel is more appropriate to this sort of classification problem.

The network architecture, represented in figure 4, is a direct extension of the ones described in (Le Cun, 1989; Le Cun et al., 1990a). The network has four hidden layers respectively named H1, H2, H3, and H4. Layers H1 and H3 are shared-weights feature extractors, while H2 and H4 are averaging/subsampling layers.

Although the size of the active part of the input is 16 by 16, the actual input is a 28 by 28 plane to avoid problems when a kernel overlaps a boundary. H1 is composed of 4 groups of 576 units arranged as 4 independent 24 by 24 feature maps. These four feature maps will be designated by H1.1, H1.2, H1.3 and H1.4. Each unit in a feature map takes its input from a 5 by 5 neighborhood on the input plane. As described above, corresponding connections on each unit in a given feature map are constrained to have the same weight. In other words, all of the 576 units in H1.1 uses the same set of 26 weights (including the bias). Of course, units in another map (say H1.4) share *another* set of 26 weights.

Layer H2 is the averaging/subsampling layer. It is composed of 4 planes of size 12 by 12. Each unit in one of these planes takes inputs on 4 units on the corresponding plane in H1. Receptive fields do not overlap. All the weights are constrained to be equal, even within a single unit. Therefore, H2 performs a local averaging and a 2 to 1 subsampling of H1 in each direction.

Layer H3 is composed of 12 feature maps. Each feature map contains 64 units arranged in a 8 by 8 plane. As before, these feature maps will be designated as H2.1, H2.2 ⋯ H2.12. The connection scheme between H2 and H3 is quite similar to the one between the input and H1, but slightly more complicated because H3 has multiple 2-D maps. Each unit receptive field is composed of one or two 5 by

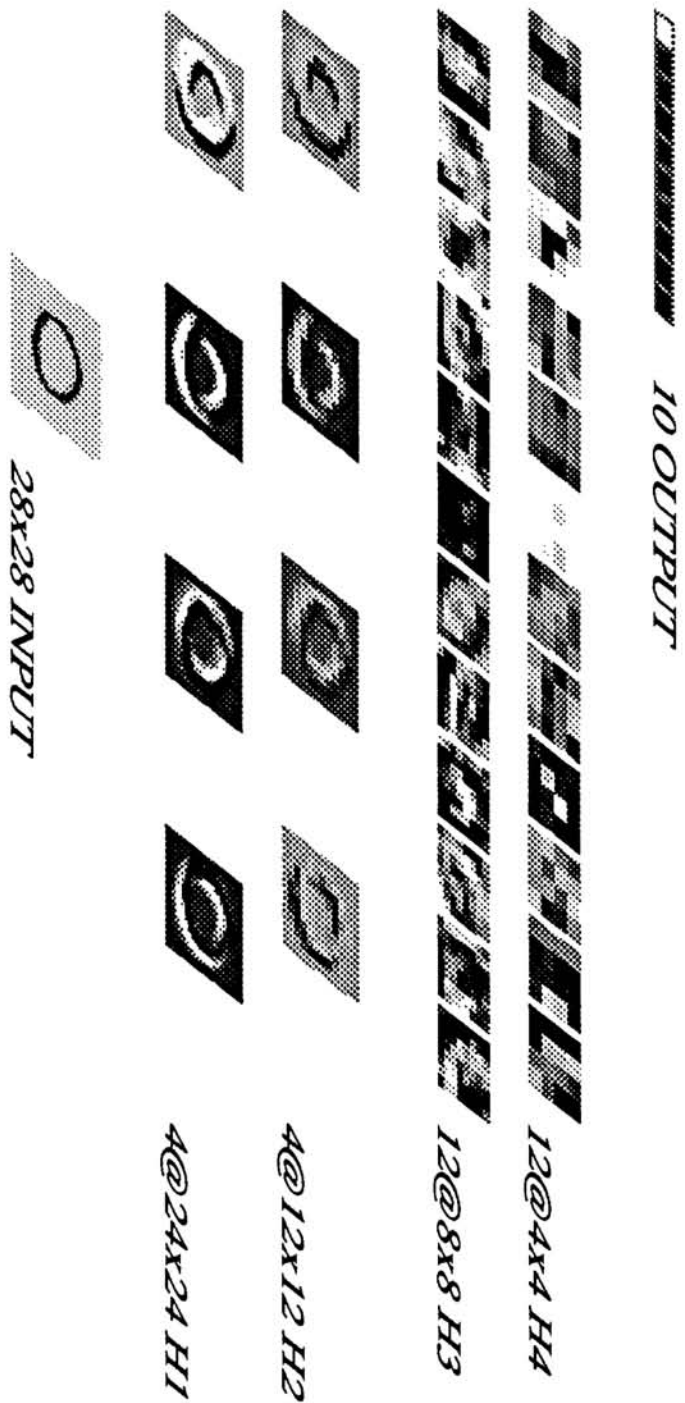

Figure 4: Network Architecture with 5 layers of fully-adaptive connections.

5 neighborhoods centered around units that are at identical positions within each H2 maps. Of course, all units in a given map are constrained to have identical weight vectors. The maps in H2 on which a map in H3 takes its inputs are chosen according to a scheme described on table 1. According to this scheme, the network is composed of two almost independent modules. Layer H4 plays the same role as layer H2, it is composed of 12 groups of 16 units arranged in 4 by 4 planes.

The output layer has 10 units and is fully connected to H4. In summary, the network has 4635 units, 98442 connections, and 2578 independent parameters. This architecture was derived using the Optimal Brain Damage technique (Le Cun et al., 1990b) starting from a previous architecture (Le Cun et al., 1990a) that had 4 times more free parameters.

## 5   RESULTS

After 30 training passes the error rate on training set (7291 handwritten plus 2549 printed digits) was 1.1% and the MSE was .017. On the whole test set (2007 handwritten plus 700 printed characters) the error rate was 3.4% and the MSE was 0.024. All the classification errors occurred on handwritten characters.

In a realistic application, the user is not so much interested in the raw error rate as in the number of rejections necessary to reach a given level of accuracy. In our case, we measured the percentage of test patterns that must be rejected in order to get 1% error rate. Our rejection criterion was based on three conditions: the activity level of the most-active output unit should by larger than a given threshold $t_1$, the activity level of the second most-active unit should be smaller than a given threshold $t_2$, and finally, the difference between the activity levels of these two units should be larger than a given threshold $t_d$. The best percentage of rejections on the complete test set was 5.7% for 1% error. On the handwritten set only, the result was 9% rejections for 1% error. It should be emphasized that the rejection thresholds were obtained using performance measures on the *test set*. About half the substitution errors in the testing set were due to faulty segmentation, and an additional quarter were due to erroneous assignment of the desired category. Some of the remaining images were ambiguous even to humans, and in a few cases the network misclassified the image for no discernible reason.

Even though a second-order version of back-propagation was used, it is interesting to note that the learning takes only 30 passes through the training set. We think this can be attributed to the large amount of redundancy present in real data. A complete training session (30 passes through the training set plus test) takes about 3 days on a SUN SPARCstation 1 using the SN2 connectionist simulator (Bottou and Le Cun, 1989).

After successful training, the network was implemented on a commercial Digital Signal Processor board containing an AT&T DSP-32C general purpose DSP chip with a peak performance of 12.5 million multiply-add operations per second on 32 bit floating point numbers. The DSP operates as a coprocessor in a PC connected to a video camera. The PC performs the digitization, binarization and segmentation

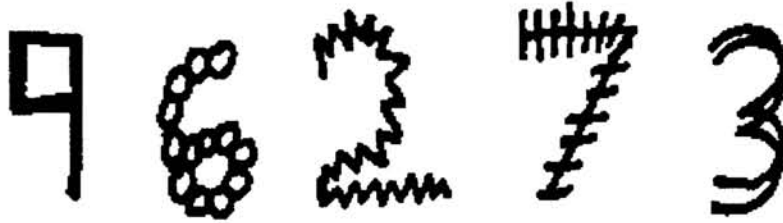

**Figure 5:** Atypical data. The network classifies these correctly, even though they are quite unlike anything in the training set.

of the image, while the DSP performs the size-normalization and the classification. The overall throughput of the digit recognizer including image acquisition is 10 to 12 classifications per second and is limited mainly by the normalization step. On normalized digits, the DSP performs more than 30 classifications per second.

## 6   CONCLUSION

Back-propagation learning was successfully applied to a large, real-world task. Our results appear to be at the state of the art in handwritten digit recognition. The network had many connections but relatively few free parameters. The network architecture and the constraints on the weights were designed to incorporate geometric knowledge about the task into the system. Because of its architecture, the network could be trained on a low-level representation of data that had minimal preprocessing (as opposed to elaborate feature extraction). Because of the redundant nature of the data and because of the constraints imposed on the network, the learning time was relatively short considering the size of the training set. Scaling properties were far better than one would expect just from extrapolating results of back-propagation on smaller, artificial problems. Preliminary results on alphanumeric characters show that the method can be directly extended to larger tasks.

The final network of connections and weights obtained by back-propagation learning was readily implementable on commercial digital signal processing hardware. Throughput rates, from camera to classified image, of more than ten digits per second were obtained.

### Acknowledgments

We thank the US Postal Service and its contractors for providing us with the zipcode database. We thank Henry Baird for useful discussions and for providing the printed-font database.

## References

Bottou, L.-Y. and Le Cun, Y. (1989). *SN2: A Simulator for Connectionist Models.* Neuristique SA, Paris, France.

Denker, J., Schwartz, D., Wittner, B., Solla, S. A., Howard, R., Jackel, L., and Hopfield, J. (1987). Large Automatic Learning, Rule Extraction and Generalization. *Complex Systems*, 1:877–922.

Denker, J. S., Gardner, W. R., Graf, H. P., Henderson, D., Howard, R. E., Hubbard, W., Jackel, L. D., Baird, H. S., and Guyon, I. (1989). Neural Network Recognizer for Hand-Written Zip Code Digits. In Touretzky, D., editor, *Neural Information Processing Systems*, volume 1, pages 323–331, Denver, 1988. Morgan Kaufmann.

Fukushima, K. and Miyake, S. (1982). Neocognitron: A new algorithm for pattern recognition tolerant of deformations and shifts in position. *Pattern Recognition*, 15:455–469.

Le Cun, Y. (1989). Generalization and Network Design Strategies. In Pfeifer, R., Schreter, Z., Fogelman, F., and Steels, L., editors, *Connectionism in Perspective*, Zurich, Switzerland. Elsevier.

Le Cun, Y., Boser, B., Denker, J. S., Henderson, D., Howard, R. E., Hubbard, W., and Jackel, L. D. (1990a). Back-Propagation Applied to Handwritten Zipcode Recognition. *Neural Computation*, 1(4).

Le Cun, Y., Denker, J. S., Solla, S., Howard, R. E. ., and Jackel, L. D. (1990b). Optimal Brain Damage. In Touretzky, D., editor, *Neural Information Processing Systems*, volume 2, Denver, 1989. Morgan Kaufman.

Patarnello, S. and Carnevali, P. (1987). Learning Networks of Neurons with Boolean Logic. *Europhysics Letters*, 4(4):503–508.

Rumelhart, D. E., Hinton, G. E., and Williams, R. J. (1986). Learning internal representations by error propagation. In *Parallel distributed processing: Explorations in the microstructure of cognition*, volume I, pages 318–362. Bradford Books, Cambridge, MA.

Tishby, N., Levin, E., and Solla, S. A. (1989). Consistent Inference of Probabilities in Layered Networks: Predictions and Generalization. In *Proceedings of the International Joint Conference on Neural Networks*, Washington DC.

Wang, C. H. and Srihari, S. N. (1988). A Framework for Object Recognition in a Visually Complex Environment and its Application to Locating Address Blocks on Mail Pieces. *International Journal of Computer Vision*, 2:125.
